# SPEECH PRODUCTION USING A NEURAL NETWORK WITH A COOPERATIVE LEARNING MECHANISM

Mitsuo Komura                Akio Tanaka

International Institute for Advanced Study of Social Information Science,
Fujitsu Limited
140 Miyamoto, Numazu-shi Shizuoka, 410-03 Japan

## ABSTRACT

We propose a new neural network model and its learning algorithm. The proposed neural network consists of four layers - input, hidden, output and final output layers. The hidden and output layers are multiple. Using the proposed SICL(Spread Pattern Information and Cooperative Learning) algorithm, it is possible to learn analog data accurately and to obtain smooth outputs. Using this neural network, we have developed a speech production system consisting of a phonemic symbol production subsystem and a speech parameter production subsystem. We have succeeded in producing natural speech waves with high accuracy.

## INTRODUCTION

Our purpose is to produce natural speech waves. In general, speech synthesis by rule is used for producing speech waves. However, there are some difficulties in speech synthesis by rule. First, the rules are very complicated. Second, extracting a generalized rule is difficult. Therefore, it is hard to synthesize a natural speech wave by using rules. We use a neural network for producing speech waves. Using a neural network, it is possible to learn speech parameters without rules. (Instead of describing rules explicitly, selecting a training data set becomes an important subject.) In this paper, we propose a new neural network model and its learning algorithm. Using the proposed neural network, it is possible to learn and produce analog data accurately. We apply the network to a speech production system and examine the system performance.

## PROPOSED NEURAL NETWORK AND ITS LEARNING ALGORITHM

We use an analog neuron-like element in a neural network. The element has a logistic activation function presented by equation (3). As a learning algorithm,

the BP(Back Propagation) method is widely used. By using this method it is possible to learn the weighting coefficients of the units whose target values are not given directly. However, there are disadvantages. First, there are singular points at 0 and 1 (outputs of the neuron-like element). Second, finding the optimum values of learning constants is not easy. We have proposed a new neural network model and its learning algorithm to solve this problem. The proposed SICL(Spread Pattern Information and Cooperative Learning) method has the following features.

(a)The singular points of the BP method are removed. (Outputs are not simply 0 or 1.) This improves the convergence rate.

(b)A spread pattern information(SI) learning algorithm is proposed. In the SI learning algorithm, the weighting coefficients from the hidden layers to the output layers are fixed to random values. Pattern information is spread over the memory space of the weighting coefficients. As a result, the network can learn analog data accurately.

(c)A cooperative learning(CL) algorithm is proposed. This algorithm makes it possible to obtain smooth and stable output. The CL system is shown in Fig.1 where $D(L)$ is a delay line which delays $L$ time units.

In the following sections, we define a three-layer network, introduce the BP method, and propose the SICL method.

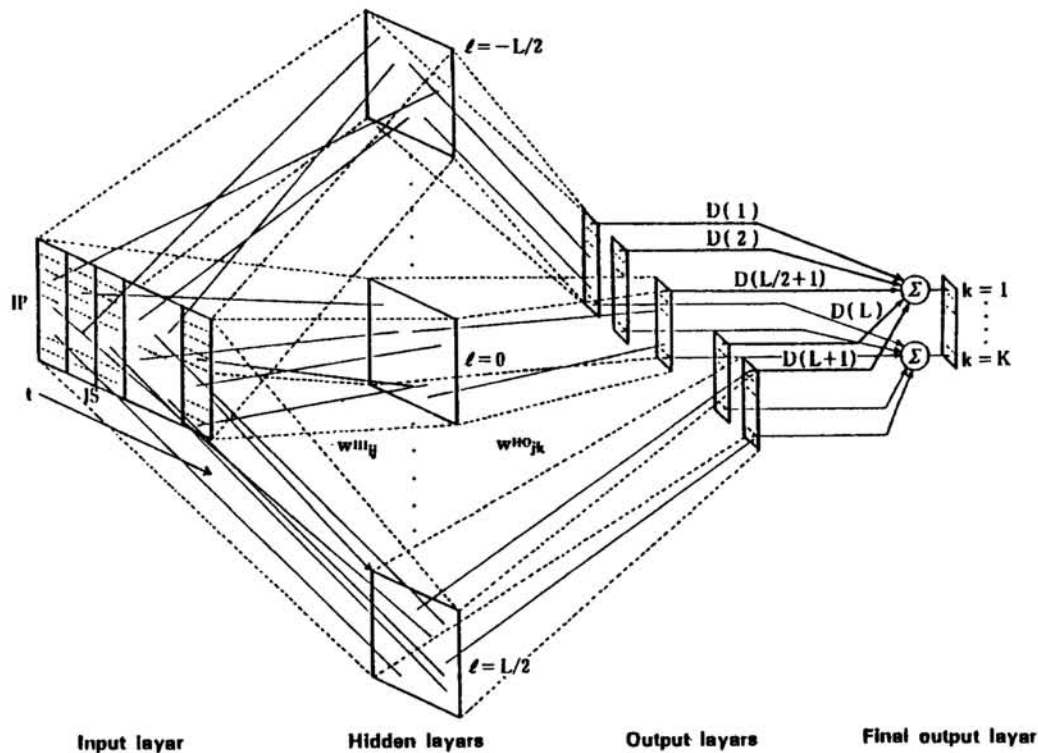

**Figure 1.** Cooperative Learning System
(Speech Parameter / Phonemic Symbol Production Subsystem)

## THREE-LAYER NETWORK

We define a three-layer network that has an input layer, a hidden layer, and an output layer. The propagation of a signal from the input layer to the hidden layer is represented as follows.

$$u_j = \Sigma_i w^{IH}_{ij} x_i, \qquad\qquad y_j = f(u_j - \theta_j), \qquad\qquad (1)$$

where $i = 1, 2, ..., I; j = 1, 2, ..., J$ and $x_i$ is an input, $y_j$ the output of the hidden layer, and $\theta_j$ a threshold value. The propagation of a signal from the hidden layer to the output layer is represented as follows.

$$v_k = \Sigma_j w^{HO}_{jk} y_j, \qquad\qquad z_k = f(v_k - \theta_k), \qquad\qquad (2)$$

where $z_k$ is the output of the output layer and $k = 1, 2, ..., K$. The activation function $f(u)$ is defined by

$$f(u) = (1 + \exp(-u + \theta))^{-1}. \qquad\qquad (3)$$

Setting $y = f(u)$, the derivation of $f(u)$ is then given by $f'(u) = y(1-y)$.

## BACK PROPAGATION (BP) METHOD

The three-layer BP algorithm is shown as follows. The back propagation error, $\delta^O_k(n)$, for an output unit is given by

$$\delta^O_k(n) = (t_k(n) - z_k(n)) f'(v_k(n)), \qquad\qquad (4)$$

where $n$ is an iteration number.

The back propagation error, $\delta^H_j(n)$, for a hidden unit is given by

$$\delta^H_j(n) = (\Sigma_k \delta^O_k(n) w^{HO}_{jk}) f'(u_j(n)). \qquad\qquad (5)$$

The change to the weight from the $i$-th to the $j$-th unit, $\Delta w^{IH}_{ij}(n)$ is given by

$$\Delta w^{IH}_{ij}(n) = a \delta^H_j(n) x_i(n) + \beta \Delta w^{IH}_{ij}(n-1). \qquad\qquad (6)$$

The change to the weight from the $j$-th to the $k$-th unit, $\Delta w^{HO}_{jk}(n)$ is given by

$$\Delta w^{HO}_{jk}(n) = a \delta^O_k(n) y_j(n) + \beta \Delta w^{HO}_{jk}(n-1), \qquad\qquad (7)$$

where $a$ and $\beta$ are learning constants, and have positive values.

## SPREAD PATTERN INFORMATION AND COOPERATIVE LEARNING (SICL) METHOD

The proposed learning algorithm - SICL method is shown as follows. The propagation of a signal from the input layer to the hidden layer is given by

$$u_j(l,n) = \Sigma_i w^{IH}_{ij}(l,n-1) x_i(n), \qquad\qquad y_j(l,n) = f(u_j(l,n) - \theta_j(l)), \qquad (8)$$

where $l$ is a stage number $(l = -L/2, ..., L/2)$. The propagation of a signal from the hidden layer to the output layer is given by

$$v_k(l,n) = \Sigma_j w^{HO}_{jk}(l) y_j(n), \qquad\qquad z_k(l,n) = f(v_k(l,n) - \theta_k(l)). \qquad (9)$$

The back propagation error, $\delta^O_k(n)$, for an output unit is given by

$$\delta O_k(l,n) = (t_k(l,n) - z_k(l,n))(z_k(l,n)(1 - z_k(l,n)) + \gamma),\qquad(10)$$

where $\gamma$ is a constant for removing singular points.

The back propagation error, $\delta^H_j(l,n)$, for a hidden unit is given by

$$\delta^H_j(l,n) = (\Sigma_k \delta O_k(l,n)\, w^{HO}_{jk}(l))(y_j(l,n)(1 - y_j(l,n)) + \gamma).\qquad(11)$$

The change to the weight from the $i$-th to the $j$-th unit, $\Delta w^{IH}_{ij}(l,n)$ is given by

$$\Delta w^{IH}_{ij}(l,n) = a\delta^H_j(l,n)x_i(n) + \beta\Delta w^{IH}_{ij}(l,n-1).\qquad(12)$$

The weight from the $j$-th to the $k$-th unit, $w^{HO}_{jk}(l)$ is given by

$$w^{HO}_{jk}(l) = c_{jk}(l),\qquad(13)$$

where $c_{jk}(l)$ is a fixed value and a random number with normal distribution. A final output is a weighting sum of outputs $z_k(l,n)$, and is given by

$$z^F_k(n) = \Sigma_l w_l z_k(l,n).\qquad(14)$$

## A SPEECH PRODUCTION SYSTEM USING THE PROPOSED NEURAL NETWORK

The block diagram of a speech production system is shown in Fig.2. The system consists of a phonemic symbol production subsystem and a speech parameter production subsystem using the proposed neural network.

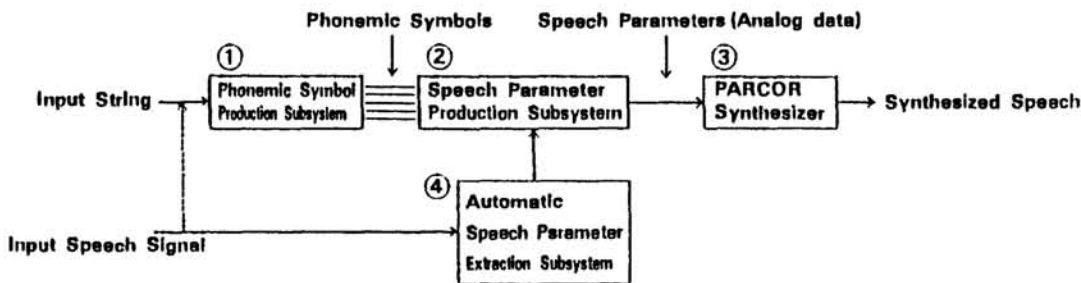

**Figure 2.** The Speech Production System Using the Proposed Neural Network

In the automatic speech parameter extraction subsystem④, speech parameters are extracted automatically from an input speech signal. Speech parameters are composed of source parameters(voiced/unvoiced ratio, pitch and source power) and a vocal tract area function(PARCOR coefficients). The extracted speech parameters are used as training data of the speech parameter production subsystem.

In the speech parameter production subsystem②, input is the string of phonemic symbols. (In the training stage, phonemic symbols are decided by the teacher using input speech data. After training, phonemic symbols are given by phonemic symbol production subsystem.) Targets are speech parameters extracted in ④.

The phonemic symbol production subsystem ① consists of a preprocessor and a learning system using the proposed neural network. In the preprocessor, a

string of input characters is converted to a string of phonemic symbols with the mean length of utterance. The input is the string of phonemic symbols converted by the preprocessor. In the training stage, the targets are actual phonemic symbols, namely, the inputs of subsystem ② which are decided by the teacher.

The output speech parameters are converted into the synthesized speech wave using the PARCOR synthesizer circuit③.

# EXPERIMENTS

We performed two separate experiments.

## Experiment I

Automatic Speech Parameter Extraction Subsystem ④
   Sampling frequency : 8 KHz        Frame length   : 20 ms
   Frame period : 10 ms

Speech Parameter Production Subsystem ②

Input layer : 1,044(36×29) units        Hidden layers : 80 units×9 stages
Output layers : 13 units×9 stages        Final output layer : 13 units

For source parameters, 3(V/UV, pitch, source power) units are assigned. For a vocal tract area function, 10(PARCOR coefficients) units are assigned. This system has an input layer, 9 hidden layers, 9 output layers, and a final output layer. For each output layer, different target data are assigned. In the final output layer, these outputs are summed with weighting coefficients.

Phonemic Symbol Production Subsystem①
   Input layer : 1,044(36×29) units    Hidden layers : 80 units×9 stages
   Output layers : 36 units×9 stages   Final output layer : 36 units

Input Speech Signal No.1

The input speech signal No.1 is 「ASAHAYAKU BANGARONI DENPOGA TODOITA 」, that is, Japanese sentence which means that "A telegram was sent to the bungalow early in the morning.". This signal is a 408 frame (4.08s) sequence.

## Experimental Results

In Fig.3, the targets and the outputs of the speech parameter production subsystem are shown. The dotted line is the sequence of target values. The solid line is the sequence of output values after the training stage. In this case, the learning constant $a$ is equal to 0.7 and $\beta$ to 0.2. After training, the actual outputs produced by SICL system agree well with the targets.

Fig.4 (a) shows the learning behaviors of the speech parameter production subsystem. In this case, the input speech signal is No.1. The learning constant $a$ is equal to 0.7 and $\beta$ to 0.2. The learning curve based on the SICL and SI

methods converged. However, the learning curve based on the BP method did not converge.

Fig.4 (b) also shows the learning behaviors. The learning constant $a$ is equal to 0.07 and $\beta$ to 0.02. In this case, all of the learning curves converged. These results show that if we use the SICL or SI method instead of the BP method, we can obtain better results.

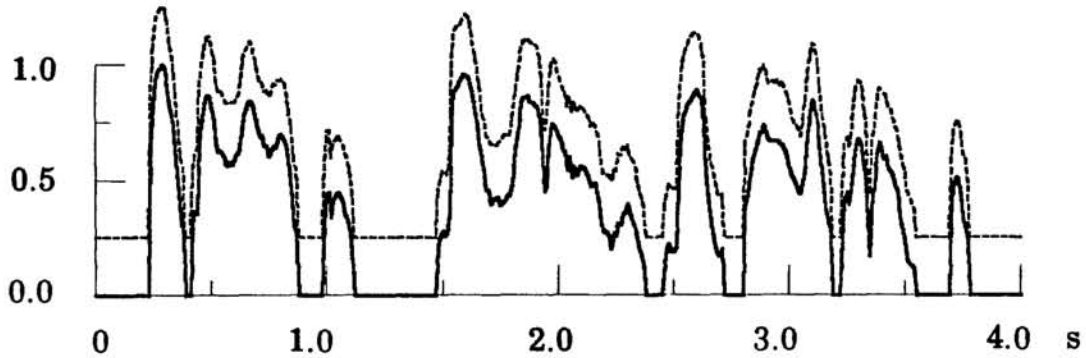

Source Power     $a = 0.7, \beta = 0.2$     Input speech signal : No.1

**Figure 3.** Targets and Outputs of Speech Parameter Production Subsystem

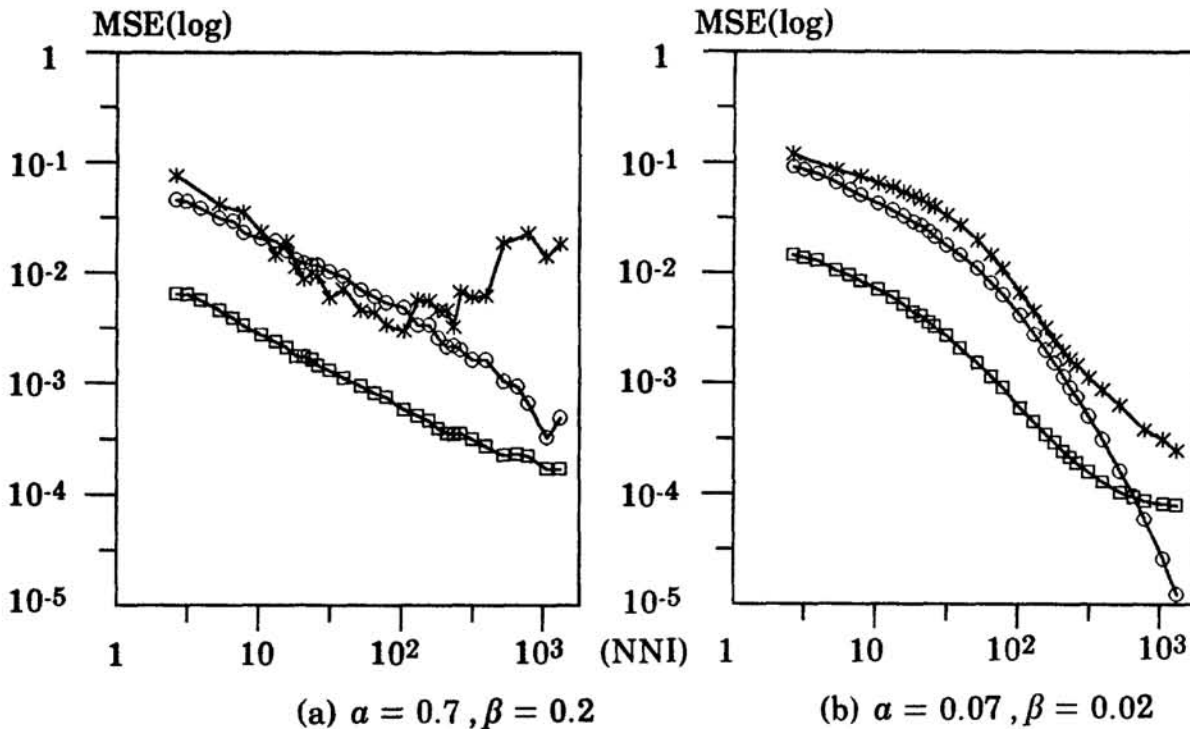

(a) $a = 0.7, \beta = 0.2$          (b) $a = 0.07, \beta = 0.02$

MSE : Mean square error          NNI : Normalized number of iteration

⊡—⊡ : SICL method (9stage)     ⊖—⊖ : SI method $(l=0)$     ★—★ : BP method

**Figure 4.** Learning Behavior of Speech Parameter Production Subsystem

Table 1 shows mean square errors of the system using SICL method for $a, \beta$. It should be noted that the domain of convergence is very wide. From these

experiments, it is seen that the SICL method almost always allows stable and smooth output to be obtained.

We also examined the system performance for another input speech signal which is a 1,608 frame (16.08 s) sequence. The learning curves converged and were similar to those for the input speech signal No.1

| $\beta$ \ $\alpha$ | 0.07 | 0.20 | 0.7 |
|---|---|---|---|
| 0.01 | $7.98 \times 10^{-5}$ | $7.82 \times 10^{-5}$ | $1.03 \times 10^{-4}$ |
| 0.02 | $7.97 \times 10^{-5}$ | $7.82 \times 10^{-5}$ | $1.09 \times 10^{-4}$ |
| 0.1 | $7.88 \times 10^{-5}$ | $7.81 \times 10^{-5}$ | $1.26 \times 10^{-4}$ |
| 0.2 | $7.85 \times 10^{-5}$ | $7.80 \times 10^{-5}$ | $1.72 \times 10^{-4}$ |
| 0.35 | $7.82 \times 10^{-5}$ | $7.81 \times 10^{-5}$ | $2.86 \times 10^{-4}$ |
| 0.7 | $7.80 \times 10^{-5}$ | $8.26 \times 10^{-5}$ | $1.72 \times 10^{-3}$ |

Input speech signal : No.1.

**TABLE 1.** Mean Square Errors for $\alpha, \beta$

## Experiment II

Speech Parameter Production Subsystem ②
Input layer : 665 (35×19)units      Hidden layers : 400 units×9 stages
Output layers : 14 (12＋2)units×9 stages

For the first target data(V/UV, power ratio, 10 PARCOR coefficients), 12 units are assigned. For the second target data(pitch, source power), 2 units are assigned.

Input Speech Signal No.2
This signal is a 13,700 frame (137s) sequence shown as follows.
/ARAARA/,/ARAIRA/,...../ARAKARA/,   /ARAKIRA/,...../ARASARA/,......
/ARARAA/,/IRARAI/,....../KARARAKA/,/KIRARAKI/,.../SARARASA/,.....

In experiment II , a learning test for producing an arbitrary combination of the input phonemes (35) was done. The learning behavior of subsystem② is shown in Fig.5. In this case, the first 12 target data items are trained. The other 2 target data items should be trained using another network. The upper line is a learning curve for the SI method, and the lower line for the SICL method. Some of the learning curves for the SI method didn't converge. However, the learning curve for the SICL method did. We can thus say that the SICL method is very powerful for actual use. (In this case, 400 hidden units is not sufficient for the size of input data. If more hidden units are used, the MSE will be small.)

## CONCLUSION

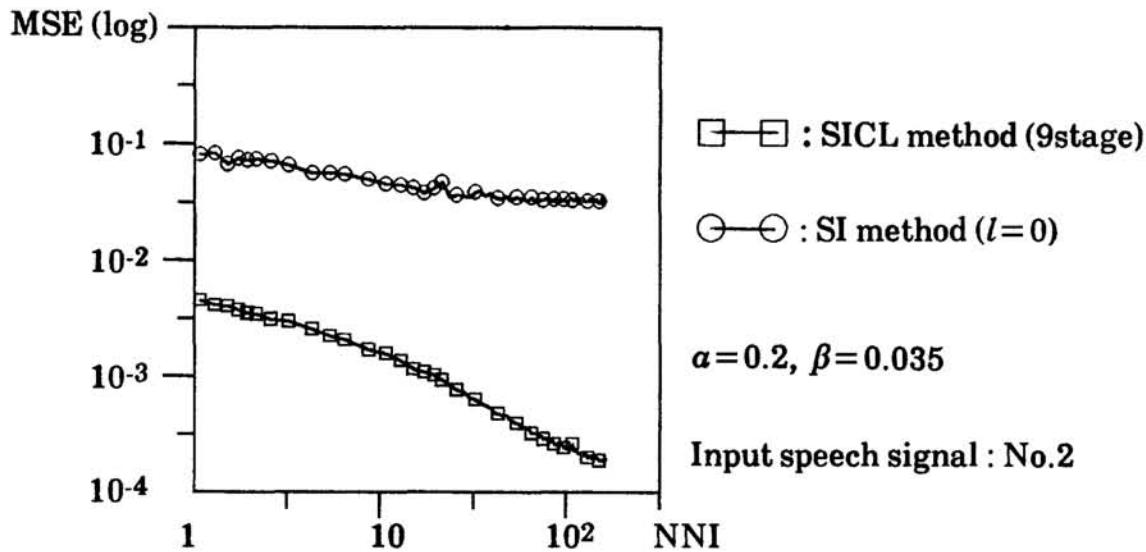

**Figure 5.** Learning Behavior of Speech Parameter Production Subsystem

From the experiments shown in previous section, it should be noted that using the SICL (Spread Pattern Information and Cooperative Learning) method makes it possible to learn speech parameters or phonemic symbols stably and to produce more natural speech waves than those synthesized by rules. If the input is the combination of a word and postpositional particle, it is easy to produce sound for unknown input data using the proposed speech production system. However, the number of the hidden units and training data will be great. Therefore we have to make the system learn phonemes among phoneme sequence. Using this training data makes it possible to produce an arbitrary sequence of phonemes. In experiment II, phonemic information (V/UV, power ratio, PARCOR coefficients) is trained. Then the range of input window was set to be 190 ms. For prosodic information (pitch, source power), we must use another network. Because, if we want to make the system learn prosodic information, we must set the range of input data wider than that of words. Using these strategies, it is possible to produce arbitrary natural speech.

## Acknowledgments

The authors thank Dr. Tosio Kitagawa, the president of IIAS-SIS for his encouragement.

## References

D. E. Rumelhart, J. L. McClelland and PDP Research Group *Parallel Distributed Processing. VOL.1* The MIT Press (1987).

T.J.Sejnowski and C.R.Rosenberg Parallel Networks that Learn to Pronounce English Text. *Complex Systems, 1, pp.145-168* (1987).

M.Komura and A.Tanaka Speech Synthesis Using a Neural Network with a Cooperative Learning Mechanism. *IEICE Tech. Rep. MBE88-8* (1988) (in Japanese).